# Reinforcement Learning using Kernel-Based Stochastic Factorization

**André M. S. Barreto**
School of Computer Science
McGill University
Montreal, Canada
amsb@cs.mcgill.ca

**Doina Precup**
School of Computer Science
McGill University
Montreal, Canada
dprecup@cs.mcgill.ca

**Joelle Pineau**
School of Computer Science
McGill University
Montreal, Canada
jpineau@cs.mcgill.ca

## Abstract

Kernel-based reinforcement-learning (KBRL) is a method for learning a decision policy from a set of sample transitions which stands out for its strong theoretical guarantees. However, the size of the approximator grows with the number of transitions, which makes the approach impractical for large problems. In this paper we introduce a novel algorithm to improve the scalability of KBRL. We resort to a special decomposition of a transition matrix, called *stochastic factorization*, to fix the size of the approximator while at the same time incorporating all the information contained in the data. The resulting algorithm, kernel-based stochastic factorization (KBSF), is much faster but still converges to a unique solution. We derive a theoretical upper bound for the distance between the value functions computed by KBRL and KBSF. The effectiveness of our method is illustrated with computational experiments on four reinforcement-learning problems, including a difficult task in which the goal is to learn a neurostimulation policy to suppress the occurrence of seizures in epileptic rat brains. We empirically demonstrate that the proposed approach is able to compress the information contained in KBRL's model. Also, on the tasks studied, KBSF outperforms two of the most prominent reinforcement-learning algorithms, namely least-squares policy iteration and fitted *Q*-iteration.

## 1 Introduction

Recent years have witnessed the emergence of several reinforcement-learning techniques that make it possible to learn a decision policy from a batch of sample transitions. Among them, Ormoneit and Sen's *kernel-based reinforcement learning* (KBRL) stands out for two reasons [1]. First, unlike other approximation schemes, KBRL always converges to a unique solution. Second, KBRL is consistent in the statistical sense, meaning that adding more data always improves the quality of the resulting policy and eventually leads to optimal performance.

Despite its nice theoretical properties, KBRL has not been widely adopted by the reinforcement learning community. One possible explanation for this is its high computational complexity. As discussed by Ormoneit and Glynn [2], KBRL can be seen as the derivation of a finite Markov decision process whose number of states coincides with the number of sample transitions collected to perform the approximation. This gives rise to a dilemma: on the one hand one wants as much data as possible to describe the dynamics of the decision problem, but on the other hand the number of transitions should be small enough to allow for the numerical solution of the resulting model.

In this paper we describe a practical way of weighting the relative importance of these two conflicting objectives. We rely on a special decomposition of a transition matrix, called *stochastic factorization*, to rewrite it as the product of two stochastic matrices of smaller dimension. As we

will see, the stochastic factorization possesses a very useful property: if we *swap* its factors, we obtain another transition matrix which retains some fundamental characteristics of the original one. We exploit this property to fix the size of KBRL's model. The resulting algorithm, *kernel-based stochastic factorization* (KBSF), is much faster than KBRL but still converges to a unique solution. We derive a theoretical bound on the distance between the value functions computed by KBRL and KBSF. We also present experiments on four reinforcement-learning domains, including the double pole-balancing task, a difficult control problem representative of a wide class of unstable dynamical systems, and a model of epileptic rat brains in which the goal is to learn a neurostimulation policy to suppress the occurrence of seizures. We empirically show that the proposed approach is able to compress the information contained in KBRL's model, outperforming both the least-squares policy iteration algorithm and fitted *Q*-iteration on the tasks studied [3, 4].

## 2  Background

The KBRL algorithm solves a continuous state-space Markov Decision Process (MDP) using a finite model approximation. A finite MDP is defined by a tuple $M \equiv (S, A, \mathbf{P}^a, \mathbf{r}^a, \gamma)$ [5]. The finite sets $S$ and $A$ are the state and action spaces. The matrix $\mathbf{P}^a \in \mathbb{R}^{|S| \times |S|}$ gives the transition probabilities associated with action $a \in A$ and the vector $\mathbf{r}^a \in \mathbb{R}^{|S|}$ stores the corresponding expected rewards. The discount factor $\gamma \in [0, 1)$ is used to give smaller weights to rewards received further in the future.

In the case of a finite MDP, we can use dynamic programming to find an optimal decision-policy $\pi^* \in A^{|S|}$ in polynomial time [5]. As well known, this is done using the concept of a *value function*. Throughout the paper, we use $\mathbf{v} \in \mathbb{R}^{|S|}$ to denote the state-value function and $\mathbf{Q} \in \mathbb{R}^{|S| \times |A|}$ to refer to the action-value function. Let the operator $\Gamma : \mathbb{R}^{|S| \times |A|} \mapsto \mathbb{R}^{|S|}$ be given by $\Gamma \mathbf{Q} = \mathbf{v}$, with $v_i = \max_j q_{ij}$, and define $\Delta : \mathbb{R}^{|S|} \mapsto \mathbb{R}^{|S| \times |A|}$ as $\Delta \mathbf{v} = \mathbf{Q}$, where the $a^{\text{th}}$ column of $\mathbf{Q}$ is given by $\mathbf{q}^a = \mathbf{r}^a + \gamma \mathbf{P}^a \mathbf{v}$. A fundamental result in dynamic programming states that, starting from $\mathbf{v}^{(0)} = \mathbf{0}$, the expression $\mathbf{v}^{(t)} = \Gamma \Delta \mathbf{v}^{(t-1)}$ gives the optimal *t*-step value function, and as $t \to \infty$ the vector $\mathbf{v}^{(t)}$ approaches $\mathbf{v}^*$, from which any optimal decision policy $\pi^*$ can be derived [5].

Consider now an MDP with continuous state space $\mathbb{S} \subset \mathbb{R}^d$ and let $S^a = \{(\mathfrak{s}_k^a, \mathfrak{r}_k^a, \hat{\mathfrak{s}}_k^a) | k = 1, 2, ..., n_a\}$ be a set of sample transitions associated with action $a \in A$, where $\mathfrak{s}_k^a, \hat{\mathfrak{s}}_k^a \in \mathbb{S}$ and $\mathfrak{r}_k^a \in \mathbb{R}$. The model constructed by KBRL has the following transition and reward functions:

$$\hat{P}^a(s_j | s_i) = \begin{cases} \kappa^a(s_i, \mathfrak{s}_k^a), & \text{if } s_j = \hat{\mathfrak{s}}_k^a, \\ 0, & \text{otherwise} \end{cases} \quad \text{and} \quad \hat{R}^a(s_i, s_j) = \begin{cases} \mathfrak{r}_k^a, & \text{if } s_j = \hat{\mathfrak{s}}_k^a, \\ 0, & \text{otherwise,} \end{cases}$$

where $\kappa^a(\cdot, \mathfrak{s}_k^a)$ is a weighting kernel centered at $\mathfrak{s}_k^a$ and defined in such a way that $\sum_{k=1}^{n_a} \kappa^a(s_i, \mathfrak{s}_k^a) = 1$ for all $s_i \in \mathbb{S}$ (for example, $\kappa^a$ can be a normalized Gaussian function; see [1] and [2] for a formal definition and other examples of valid kernels). Since only transitions ending in the states $\hat{\mathfrak{s}}_k^a$ have a non-zero probability of occurrence, one can solve a finite MDP $\hat{M}$ whose space is composed solely of these $n = \sum_a n_a$ states [2, 6]. After the optimal value function of $\hat{M}$ has been found, the value of any state $s_i \in \mathbb{S}$ can be computed as $Q(s_i, a) = \sum_{k=1}^{n_a} \kappa^a(s_i, \mathfrak{s}_k^a) \left[ \mathfrak{r}_k^a + \gamma \hat{V}^*(\hat{\mathfrak{s}}_k^a) \right]$. Ormoneit and Sen [1] proved that, if $n_a \to \infty$ for all $a \in A$ and the widths of the kernels $\kappa^a$ shrink at an "admissible" rate, the probability of choosing a suboptimal action based on $Q(s_i, a)$ converges to zero.

As discussed in the introduction, the problem with the practical application of KBRL is that, as $n$ increases, so does the cost of solving the MDP derived by this algorithm. To alleviate this problem, Jong and Stone [6] propose growing incrementally the set of sample transitions, using a prioritized sweeping approach to guide the exploration of the state space. In this paper we present a new method for addressing this problem, using stochastic factorization.

## 3  Stochastic factorization

A stochastic matrix has only non-negative elements and each of its rows sums to 1. That said, we can introduce the concept that will serve as a cornerstone for the rest of the paper:

**Definition 1** *Given a stochastic matrix* $\mathbf{P} \in \mathbb{R}^{n \times p}$, *the relation* $\mathbf{P} = \mathbf{DK}$ *is called a* stochastic factorization *of* $\mathbf{P}$ *if* $\mathbf{D} \in \mathbb{R}^{n \times m}$ *and* $\mathbf{K} \in \mathbb{R}^{m \times p}$ *are also stochastic matrices. The integer* $m > 0$ *is the* order *of the factorization.*

This mathematical concept has been explored before. For example, Cohen and Rothblum [7] briefly discuss it as a special case of non-negative matrix factorization, while Cutler and Breiman [8] focus on slightly modified versions of the stochastic factorization for statistical data analysis. However, in this paper we will focus on a useful property of this type of factorization that seems to have passed unnoticed thus far. We call it the "*stochastic-factorization trick*":

> *Given a stochastic factorization of a square matrix,* $\mathbf{P} = \mathbf{DK}$, *swapping the factors of the factorization yields another transition matrix* $\bar{\mathbf{P}} = \mathbf{KD}$, *potentially much smaller than the original, which retains the basic topology and properties of* $\mathbf{P}$.

The stochasticity of $\bar{\mathbf{P}}$ follows immediately from the same property of $\mathbf{D}$ and $\mathbf{K}$. What is perhaps more surprising is the fact that this matrix shares some fundamental characteristics with the original matrix $\mathbf{P}$. Specifically, it is possible to show that: (*i*) for each recurrent class in $\mathbf{P}$ there is a corresponding class in $\bar{\mathbf{P}}$ with the same period and, given some simple assumptions about the factorization, (*ii*) $\mathbf{P}$ is irreducible if and only if $\bar{\mathbf{P}}$ is irreducible and (*iii*) $\mathbf{P}$ is regular if and only if $\bar{\mathbf{P}}$ is regular (see [9] for details).

Given the strong connection between $\mathbf{P} \in \mathbb{R}^{n \times n}$ and $\bar{\mathbf{P}} \in \mathbb{R}^{m \times m}$, the idea of replacing the former by the latter comes almost inevitably. The motivation for this would be, of course, to save computational resources when $m < n$. In this paper we are interested in exploiting the stochastic-factorization trick to reduce the computational cost of dynamic programming. The idea is straightforward: given stochastic factorizations of the transition matrices $\mathbf{P}^a$, we can apply our trick to obtain a reduced MDP that will be solved in place of the original one. In the most general scenario, we would have one independent factorization $\mathbf{P}^a = \mathbf{D}^a \mathbf{K}^a$ for each action $a \in A$. However, in the current work we will focus on a particular case which will prove to be convenient both mathematically and computationally. When all factorizations share the same matrix $\mathbf{D}$, it is easy to derive theoretical guarantees regarding the quality of the solution of the reduced MDP:

**Proposition 1** *Let* $M \equiv (S, A, \mathbf{P}^a, \mathbf{r}^a, \gamma)$ *be a finite MDP with* $|S| = n$ *and* $0 \leq \gamma < 1$. *Let* $\mathbf{DK}^a = \mathbf{P}^a$ *be* $|A|$ *stochastic factorizations of order m and let* $\bar{\mathbf{r}}^a$ *be vectors in* $\mathbb{R}^m$ *such that* $\mathbf{D}\bar{\mathbf{r}}^a = \mathbf{r}^a$ *for all* $a \in A$. *Define the MDP* $\bar{M} \equiv (\bar{S}, A, \bar{\mathbf{P}}^a, \bar{\mathbf{r}}^a, \gamma)$, *with* $|\bar{S}| = m$ *and* $\bar{\mathbf{P}}^a = \mathbf{K}^a \mathbf{D}$. *Then,*

$$\|\mathbf{v}^* - \tilde{\mathbf{v}}\|_\infty \leq \frac{\bar{C}}{(1-\gamma)^2} \max_i \left(1 - \max_j d_{ij}\right), \qquad (1)$$

*where* $\tilde{\mathbf{v}} = \Gamma \mathbf{D} \bar{\mathbf{Q}}^*$, $\bar{C} = \max_{a,k} \bar{r}_k^a - \min_{a,k} \bar{r}_k^a$, *and* $\|\cdot\|_\infty$ *is the maximum norm.*

**Proof.** Since $\mathbf{r}^a = \mathbf{D}\bar{\mathbf{r}}^a$ and $\mathbf{D}\bar{\mathbf{P}}^a = \mathbf{DK}^a \mathbf{D} = \mathbf{P}^a \mathbf{D}$ for all $a \in A$, the stochastic matrix $\mathbf{D}$ satisfies Sorg and Singh's definition of a *soft homomorphism* between $M$ and $\bar{M}$ (see equations (25)–(28) in [10]). Applying Theorem 1 by the same authors, we know that $\left\|\Gamma(\mathbf{Q}^* - \mathbf{D}\bar{\mathbf{Q}}^*)\right\|_\infty \leq (1-\gamma)^{-1} \sup_{i,t}(1 - \max_j d_{ij})\bar{\delta}_i^{(t)}$, where $\bar{\delta}_i^{(t)} = \max_{j:d_{ij}>0,k} \bar{q}_{jk}^{(t)} - \min_{j:d_{ij}>0,k} \bar{q}_{jk}^{(t)}$ and $\bar{q}_{jk}^{(t)}$ are elements of the optimal $t$-step action-value function of $\bar{M}$, $\bar{\mathbf{Q}}^{(t)} = \Delta \bar{\mathbf{v}}^{(t-1)}$. In order to obtain our bound, we note that $\left\|\Gamma \mathbf{Q}^* - \Gamma \mathbf{D}\bar{\mathbf{Q}}^*\right\|_\infty \leq \left\|\Gamma(\mathbf{Q}^* - \mathbf{D}\bar{\mathbf{Q}}^*)\right\|_\infty$ and, for all $t > 0$, $\bar{\delta}_i^{(t)} \leq (1-\gamma)^{-1}(\max_{a,k} \bar{r}_k^a - \min_{a,k} \bar{r}_k^a)$. □

Proposition 1 elucidates the basic mechanism through which one can exploit the stochastic-factorization trick to reduce the number of states in an MDP. However, in order to apply this idea in practice, one must actually *compute* the factorizations. This computation can be expensive, exceeding the computational effort necessary to calculate $\mathbf{v}^*$ [11, 9]. In the next section we discuss a strategy to reduce the computational cost of the proposed approach.

## 4 Kernel-based stochastic factorization

In Section 2 we presented KBRL, an approximation scheme for reinforcement learning whose main drawback is its high computational complexity. In Section 3 we discussed how the stochastic-factorization trick can in principle be useful to reduce an MDP, as long as one circumvents the computational burden imposed by the calculation of the matrices involved in the process. We now show how to leverage these two components to produce an algorithm called *kernel-based stochastic factorization* (KBSF) that overcomes these computational limitations.

As outlined in Section 2, KBRL defines the probability of a transition from state $\hat{\mathfrak{s}}_i^b$ to state $\hat{\mathfrak{s}}_k^a$ via kernel-averaging, formally denoted $\kappa^a(\hat{\mathfrak{s}}_i^b, \mathfrak{s}_k^a)$, where $a, b \in A$. So for each action $a \in A$, the state $\hat{\mathfrak{s}}_i^b$ has an associated stochastic vector $\hat{\mathbf{p}}_j^a \in \mathbb{R}^{1 \times n}$ whose non-zero entries correspond to the function $\kappa^a(\hat{\mathfrak{s}}_i^b, \cdot)$ evaluated at $\mathfrak{s}_k^a, k = 1, 2, \ldots, n_a$. Recall that we are dealing with a continuous state space, so it is possible to compute an analogous vector for any $s_i \in \mathbb{S}$. Therefore, we can link each state of the original MDP with $|A|$ $n$-dimensional stochastic vectors. The core strategy of KBSF is to find a set of $m$ representative states associated with vectors $\mathbf{k}_i^a \in \mathbb{R}^{1 \times n}$ whose convex combination can approximate the rows of the corresponding $\hat{\mathbf{P}}^a$.

KBRL's matrices $\hat{\mathbf{P}}^a$ have a very specific structure, since only transitions ending in states $\hat{\mathfrak{s}}_i^a$ associated with action $a$ have a non-zero probability of occurrence. Suppose now we want to apply the stochastic-factorization trick to KBRL's MDP. Assuming that the matrices $\mathbf{K}^a$ have the same structure as $\hat{\mathbf{P}}^a$, when computing $\bar{\mathbf{P}}^a = \mathbf{K}^a \mathbf{D}$ we only have to look at the submatrices of $\mathbf{K}^a$ and $\mathbf{D}$ corresponding to the $n_a$ non-zero columns of $\mathbf{K}^a$. We call these matrices $\dot{\mathbf{K}}^a \in \mathbb{R}^{m \times n_a}$ and $\dot{\mathbf{D}}^a \in \mathbb{R}^{n_a \times m}$.

Let $\{\bar{s}_1, \bar{s}_2, \ldots, \bar{s}_m\}$ be a set of representative states in $\mathbb{S}$. KBSF computes matrices $\dot{\mathbf{D}}^a$ and $\dot{\mathbf{K}}^a$ with elements $\dot{d}_{ij}^a = \bar{\kappa}(\hat{\mathfrak{s}}_i^a, \bar{s}_j)$ and $\dot{k}_{ij}^a = \kappa^a(\bar{s}_i, \mathfrak{s}_j^a)$, where $\bar{\kappa}$ is also a kernel. Obviously, once we have $\dot{\mathbf{D}}^a$ and $\dot{\mathbf{K}}^a$ it is trivial to compute $\mathbf{D}$ and $\mathbf{K}^a$. Depending on how the states $\bar{s}_i$ and the kernels $\bar{\kappa}$ are defined, we have $\mathbf{DK}^a \approx \hat{\mathbf{P}}^a$ for all $a \in A$. The important point here is that the matrices $\mathbf{P}^a = \mathbf{DK}^a$ are never actually computed, but instead we solve an MDP with $m$ states whose dynamics are given by $\bar{\mathbf{P}}^a = \mathbf{K}^a \mathbf{D} = \dot{\mathbf{K}}^a \dot{\mathbf{D}}^a$. Algorithm 1 gives a step-by-step description of KBSF.

---

**Algorithm 1** KBSF

> **Input:** $S^a$ for all $a \in A$, $m$
> Select a set of representative states $\{\bar{s}_1, \bar{s}_2, \ldots, \bar{s}_m\}$
> **for** each $a \in A$ **do**
>     Compute matrix $\dot{\mathbf{D}}^a$: $\dot{d}_{ij}^a = \bar{\kappa}(\hat{\mathfrak{s}}_i^a, \bar{s}_j)$
>     Compute matrix $\dot{\mathbf{K}}^a$: $\dot{k}_{ij}^a = \kappa^a(\bar{s}_i, \mathfrak{s}_j^a)$
>     Compute vector $\bar{\mathbf{r}}^a$: $\bar{r}_i^a = \sum_j \dot{k}_{ij}^a \mathfrak{r}_j^a$
> **end for**
> Solve $\bar{M} \equiv (\bar{S}, A, \bar{\mathbf{P}}^a, \bar{\mathbf{r}}^a, \gamma)$, with $\bar{\mathbf{P}}^a = \dot{\mathbf{K}}^a \dot{\mathbf{D}}^a$
> Return $\tilde{\mathbf{v}} = \Gamma \mathbf{D} \bar{\mathbf{Q}}^*$, where $\mathbf{D}^{\mathsf{T}} = \left[ \left(\dot{\mathbf{D}}^{a_1}\right)^{\mathsf{T}} \left(\dot{\mathbf{D}}^{a_2}\right)^{\mathsf{T}} \ldots \left(\dot{\mathbf{D}}^{a_{|A|}}\right)^{\mathsf{T}} \right]$

---

Observe that we did not describe how to define the representative states $\bar{s}_i$. Ideally, these states would be linked to vectors $\mathbf{k}_i^a$ forming a convex hull which contains the rows of $\hat{\mathbf{P}}^a$. In practice, we can often resort to simple methods to pick states $\bar{s}_i$ in strategic regions of $\mathbb{S}$. In Section 5 we give an example of how to do so. Also, the reader might have noticed that the stochastic factorizations computed by KBSF are in fact approximations of the matrices $\hat{\mathbf{P}}^a$. The following proposition extends the result of the previous section to the approximate case:

**Proposition 2** *Let $\hat{M} \equiv (S, A, \hat{\mathbf{P}}^a, \hat{\mathbf{r}}^a, \gamma)$ be the finite MDP derived by KBRL and let $\mathbf{D}$, $\mathbf{K}^a$, and $\bar{\mathbf{r}}^a$ be the matrices and vector computed by KBSF. Then,*

$$\|\hat{\mathbf{v}}^* - \tilde{\mathbf{v}}\|_\infty \leq \frac{1}{1-\gamma} \max_a \|\hat{\mathbf{r}}^a - \mathbf{D}\bar{\mathbf{r}}^a\|_\infty + \frac{1}{(1-\gamma)^2} \left( \bar{C} \max_i \left(1 - \max_j d_{ij}\right) + \frac{\hat{C}\gamma}{2} \max_a \|\hat{\mathbf{P}}^a - \mathbf{DK}^a\|_\infty \right), \quad (2)$$

*where $\hat{C} = \max_{a,i} \hat{r}_i^a - \min_{a,i} \hat{r}_i^a$.*

**Proof.** Let $M \equiv (S, A, \mathbf{DK}^a, \mathbf{D}\bar{\mathbf{r}}^a, \gamma)$. It is obvious that

$$\|\hat{\mathbf{v}}^* - \tilde{\mathbf{v}}\|_\infty \leq \|\hat{\mathbf{v}}^* - \mathbf{v}^*\|_\infty + \|\mathbf{v}^* - \tilde{\mathbf{v}}\|_\infty. \quad (3)$$

In order to provide a bound for $\|\hat{\mathbf{v}}^* - \mathbf{v}^*\|_\infty$, we apply Whitt's Theorem 3.1 and Corollary (b) of his Theorem 6.1 [12], with all mappings between $\hat{M}$ and $M$ taken to be identities, to obtain

$$\|\hat{\mathbf{v}}^* - \mathbf{v}^*\|_\infty \leq \frac{1}{1-\gamma} \left( \max_a \|\hat{\mathbf{r}}^a - \mathbf{D}\bar{\mathbf{r}}^a\|_\infty + \frac{\hat{C}\gamma}{2(1-\gamma)} \max_a \|\hat{\mathbf{P}}^a - \mathbf{DK}^a\|_\infty \right). \quad (4)$$

Resorting to Proposition 1, we can substitute (1) and (4) in (3) to obtain (2). $\square$

Notice that when **D** is deterministic—that is, when all its non-zero elements are 1—expression (2) reduces to Whitt's classical result regarding state aggregation in dynamic programming [12]. On the other hand, when the stochastic factorizations are exact, we recover (1), which is a computable version of Sorg and Singh's bound for soft homomorphisms [10]. Finally, if we have exact deterministic factorizations, the right-hand side of (2) reduces to zero. This also makes sense, since in this case the stochastic-factorization trick gives rise to an exact homomorphism [13].

As shown in Algorithm 1, KBSF is very simple to understand and to implement. It is also fast, requiring only $O(nm^2|A|)$ operations to build a reduced version of an MDP. Finally, and perhaps most importantly, KBSF always converges to a unique solution whose distance to the optimal one is bounded. In the next section we show how all these qualities turn into practical benefits.

## 5 Experiments

We now present a series of computational experiments designed to illustrate the behavior of KBSF in a variety of challenging domains. We start with a simple problem showing that KBSF is indeed capable of compressing the information contained in KBRL's model. We then move to more difficult tasks, and compare KBSF with other state-of-the-art reinforcement-learning algorithms.

All problems considered in this section have a continuous state space and a finite number of actions and were modeled as discounted tasks with $\gamma = 0.99$. The algorithms's results correspond to the performance of the greedy decision policy derived from the final value function computed. In all cases, the decision policies were evaluated on a set of test states from which the tasks cannot be easily solved. This makes the tasks considerably harder, since the algorithms must provide a good approximation of the value function over a larger region of the state space.

The experiments were carried out in the same way for all tasks: first, we collected a set of $n$ sample transitions $(\mathfrak{s}_k^a, \mathfrak{r}_k^a, \hat{\mathfrak{s}}_k^a)$ using a uniformly-random exploration policy. Then the states $\hat{\mathfrak{s}}_k^a$ were grouped by the $k$-means algorithm into $m$ clusters and a Gaussian kernel $\bar{\kappa}$ was positioned at the center of each resulting cluster [14]. These kernels defined the models used by KBSF to approximate the value function. This process was repeated 50 times for each task.

We adopted the same width for all kernels. The algorithms were executed on each task with the following values for this parameter: $\{1, 0.1, 0.01\}$. The results reported represent the best performance of the algorithms over the 50 runs; that is, for each $n$ and each $m$ we picked the width that generated the maximum average return. Throughout this section we use the following convention to refer to specific instances of each method: the first number enclosed in parentheses after an algorithm's name is $n$, the number of sample transitions used in the approximation, and the second one is $m$, the size of the model used to approximate the value function. Note that for KBRL $n$ and $m$ coincide.

Figure 1 shows the results obtained by KBRL and KBSF on the puddle-world task [15]. In Figure 1a and 1b we observe the effect of fixing the number of transitions $n$ and varying the number of representative states $m$. As expected, the results of KBSF improve as $m \rightarrow n$. More surprising is the fact that KBSF has essentially the same performance as KBRL using models one order of magnitude smaller. This indicates that KBSF is summarizing well the information contained in the data. Depending on the values of $n$ and $m$, this compression may represent a significant reduction of computational resources. For example, by replacing KBRL(8000) with KBSF(8000, 90), we obtain a decrease of more than 99% on the number of operations performed to find a policy, as shown in Figure 1b (the cost of constructing KBSF's MDP is included in all reported run times).

In Figures 1c and 1d we fix $m$ and vary $n$. Observe in Figure 1c how KBRL and KBSF have similar performances, and both improve as $n \rightarrow \infty$. However, since KBSF is using a model of fixed size, its computational cost depends only linearly on $n$, whereas KBRL's cost grows with $n^3$. This explains the huge difference in the algorithms's run times shown in Figure 1d.

Next we evaluate how KBSF compares to other reinforcement-learning approaches. We first contrast our method with Lagoudakis and Parr's least-squares policy iteration algorithm (LSPI) [3]. LSPI is a natural candidate here because it also builds an approximator of fixed size out of a batch of sample transitions. In all experiments LSPI used the same data and approximation architectures as KBSF (to be fair, we fixed the width of KBSF's kernel $\kappa^a$ at 1 in the comparisons).

Figure 2 shows the results of LSPI and KBSF on the single and double pole-balancing tasks [16]. We call attention to the fact that the version of the problems used here is significantly harder than

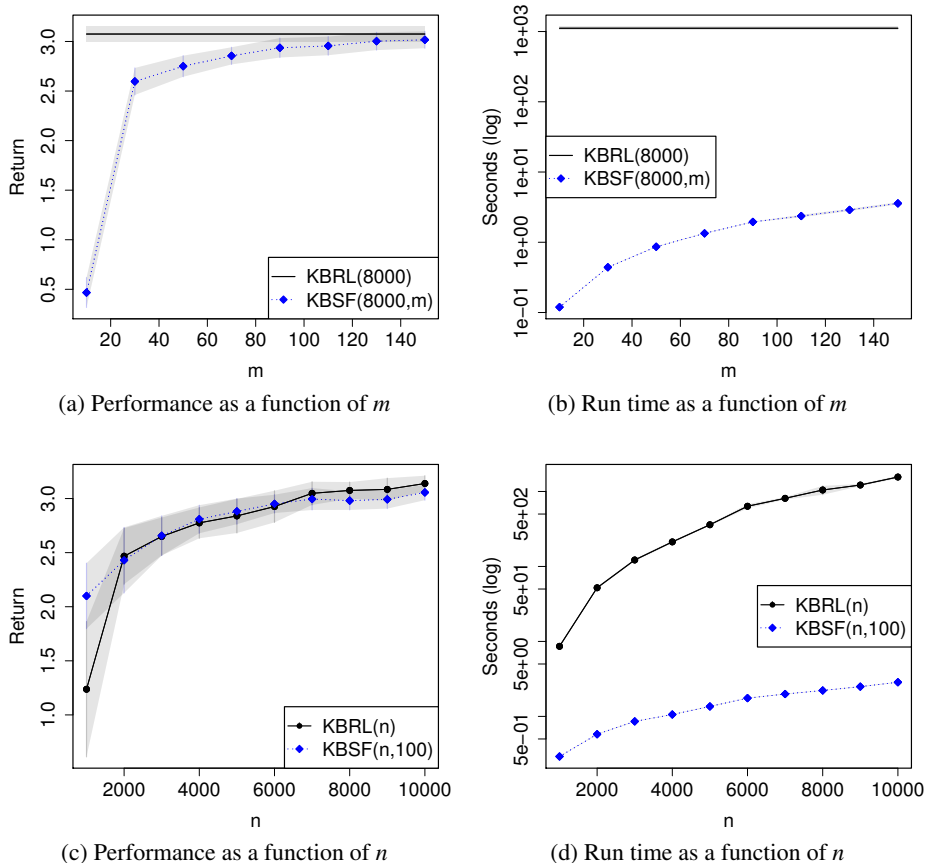

(a) Performance as a function of $m$       (b) Run time as a function of $m$

(c) Performance as a function of $n$       (d) Run time as a function of $n$

Figure 1: Results on the puddle-world task averaged over 50 runs. The algorithms were evaluated on two sets of states distributed over the region of the state space surrounding the "puddles". The first set was a $3 \times 3$ grid over $[0.1, 0.3] \times [0.3, 0.5]$ and the second one was composed of four states: $\{0.1, 0.3\} \times \{0.9, 1.0\}$. The shadowed regions represent 99% confidence intervals.

the more commonly-used variants in which the decision policies are evaluated on a single state close to the origin. This is probably the reason why LSPI achieves a success rate of no more than 60% on the single pole-balancing task, as shown in Figure 2a. In contrast, KBSF's decision policies are able to balance the pole in 90% of the attempts, on average, using as few as $m = 30$ representative states.

The results of KBSF on the double pole-balancing task are still more impressive. As Wieland [17] rightly points out, this version of the problem is considerably more difficult than its single pole variant, and previous attempts to apply reinforcement-learning techniques to this domain resulted in disappointing performance [18]. As shown in Figure 2c, KBSF($10^6$, 200) is able to achieve a success rate of more than 80%. To put this number in perspective, recall that some of the test states are quite challenging, with the two poles inclined and falling in opposite directions.

The good performance of KBSF comes at a relatively low computational cost. A conservative estimate reveals that, were KBRL($10^6$) run on the same computer used for these experiments, we would have to wait for more than 6 *months* to see the results. KBSF($10^6$, 200) delivers a decision policy in less than 7 minutes. KBSF's computational cost also compares well with that of LSPI, as shown in Figures 2b and 2d. LSPI's policy-evaluation step involves the update and solution of a linear system of equations, which take $O(nm^2)$ and $O(m^3|A|^3)$, respectively. In addition, the policy-update stage requires the definition of $\pi(\hat{\mathfrak{s}}_k^a)$ for all $n$ states in the set of sample transitions. In contrast, KBSF only performs $O(m^3)$ operations to evaluate a decision policy and $O(m^2|A|)$ operations to update it.

We conclude our empirical evaluation of KBSF by using it to learn a neurostimulation policy for the treatment of epilepsy. In order to do so, we use a generative model developed by Bush et al. [19] based on real data collected from epileptic rat hippocampus slices. This model was shown to re-

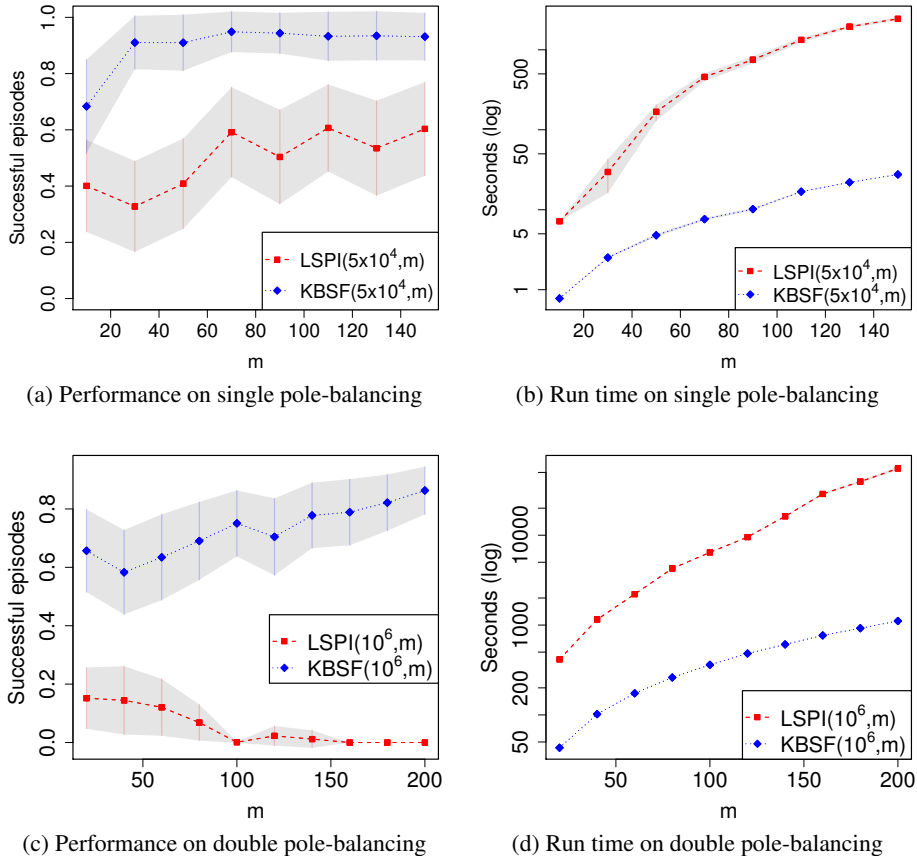

| | |
|---|---|
| (a) Performance on single pole-balancing | (b) Run time on single pole-balancing |
| (c) Performance on double pole-balancing | (d) Run time on double pole-balancing |

Figure 2: Results on the pole-balancing tasks averaged over 50 runs. The values correspond to the fraction of episodes initiated from the test states in which the pole(s) could be balanced for 3000 steps (one minute of simulated time). The test sets were regular grids defined over the hypercube centered at the origin and covering 50% of the state-space axes in each dimension (we used a resolution of 3 and 2 cells per dimension for the single and double versions of the problem, respectively). Shadowed regions represent 99% confidence intervals.

produce the seizure pattern of the original dynamical system and was later validated through the deployment of a learned treatment policy on a real brain slice [20]. The associated decision problem has a five-dimensional continuous state space and extremely non-linear dynamics. At each state the agent must choose whether or not to apply an electrical pulse. The goal is to suppress seizures while minimizing the total amount of stimulation needed to do so.

We use as a baseline for our comparisons the fixed-frequency stimulation policies usually adopted in standard *in vitro* clinical studies [20]. In particular, we considered policies that apply electrical pulses at frequencies of 0 Hz, 0.5 Hz, 1 Hz, and 1.5 Hz. For this task we ran LSPI and KBSF with sparse kernels, that is, we only computed the value of the Gaussian function at the 6-nearest neighbors of a given state (thus defining a simplex with the same dimension as the state space). This modification made it possible to use $m = 50,000$ representative states with KBSF. Since for LSPI the reduction on the computational cost was not very significant, we fixed $m = 50$ to keep its run time within reasonable bounds.

We compare the decision policies returned by KBSF and LSPI with those computed by fitted $Q$-iteration using Ernst et al.'s extra-trees algorithm [4]. This approach has shown excellent performance on several reinforcement-learning tasks [4]. We used the extra-trees algorithm to build an ensemble of 30 trees. The algorithm was run for 50 iterations, with the structure of the trees fixed after the 10th one. The number of cut-directions evaluated at each node was fixed at $\dim(S) = 5$, and the minimum number of elements required to split a node, denoted here by $\eta_{\min}$, was selected from the set $\{20, 30, ..., 50, 100, 150, ..., 200\}$. In general, we observed that the performance of the tree-

based method improved with smaller values for $\eta_{\min}$, with an expected increase in the computational cost. Thus, in order to give an overall characterization of the performance of fitted $Q$-iteration, we report the results obtained with the extreme values of $\eta_{\min}$. The respective instances of the tree-based approach are referred to as T20 and T200.

Figure 3 shows the results on the epilepsy-suppression task. In order to obtain different compromises of the problem's two conflicting objectives, we varied the relative magnitude of the penalties associated with the occurrence of seizures and with the application of an electrical pulse [19, 20]. In particular, we fixed the latter at $-1$ and varied the former with values in $\{-10, -20, -40\}$. This appears in the plots as subscripts next to the algorithms's names. As shown in Figure 3a, LSPI's policies seem to prioritize reduction of stimulation at the expense of higher seizure occurrence, which is clearly sub-optimal from a clinical point of view. T200 also performs poorly, with solutions representing no advance over the fixed-frequency stimulation strategies. In contrast, T20 and KBSF are both able to generate decision policies superior to the 1 Hz policy, which is the most efficient stimulation regime known to date in the clinical literature [21]. However, as shown in Figure 3b, KBSF is able to do it at least 100 times faster than the tree-based method.

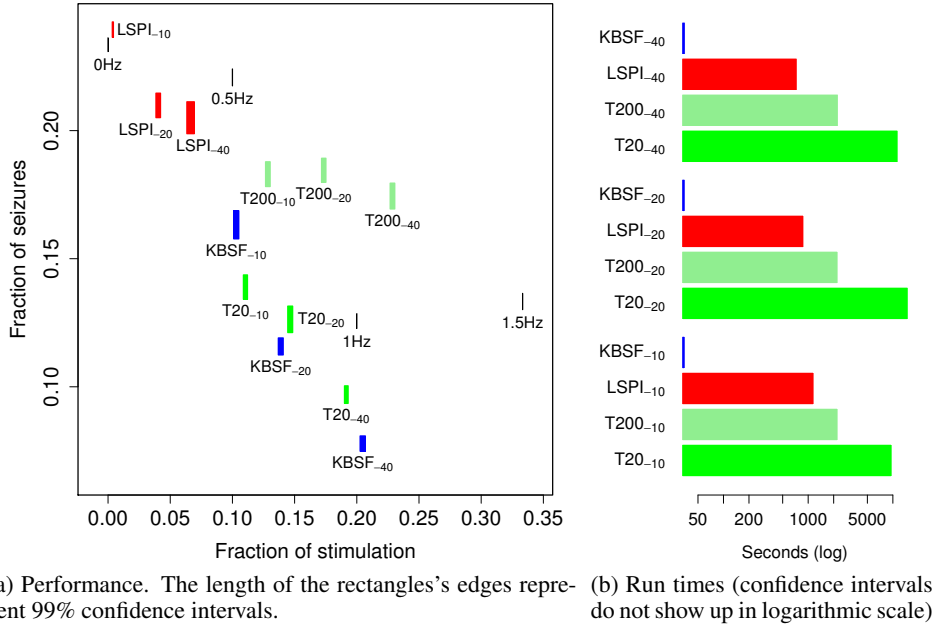

(a) Performance. The length of the rectangles's edges represent 99% confidence intervals.

(b) Run times (confidence intervals do not show up in logarithmic scale)

Figure 3: Results on the epilepsy-suppression problem averaged over 50 runs. The algorithms used $n = 500,000$ sample transitions to build the approximations. The decision policies were evaluated on episodes of $10^5$ transitions starting from a fixed set of 10 test states drawn uniformly at random.

## 6   Conclusions

We presented KBSF, a reinforcement-learning algorithm that emerges from the application of the stochastic-factorization trick to KBRL. As discussed, our algorithm is simple, fast, has good theoretical guarantees, and always converges to a unique solution. Our empirical results show that KBSF is able to learn very good decision policies with relatively low computational cost. It also has predictable behavior, generally improving its performance as the number of sample transitions or the size of its approximation model increases. In the future, we intend to investigate more principled strategies to select the representative states, based on the large body of literature available on kernel methods. We also plan to extend KBSF to the on-line scenario, where the intermediate decision policies generated during the learning process guide the collection of new sample transitions.

**Acknowledgments**

The authors would like to thank Keith Bush for making the epilepsy simulator available and also Yuri Grinberg for helpful discussions regarding this work. Funding for this research was provided by the National Institutes of Health (grant R21 DA019800) and the NSERC Discovery Grant program.

# References

[1] D. Ormoneit and S. Sen. Kernel-based reinforcement learning. *Machine Learning*, 49 (2–3): 161–178, 2002.

[2] D. Ormoneit and P. Glynn. Kernel-based reinforcement learning in average-cost problems. *IEEE Transactions on Automatic Control*, 47(10):1624–1636, 2002.

[3] M. G. Lagoudakis and R. Parr. Least-squares policy iteration. *Journal of Machine Learning Research*, 4:1107–1149, 2003.

[4] D. Ernst, P. Geurts, and L. Wehenkel. Tree-based batch mode reinforcement learning. *Journal of Machine Learning Research*, 6:503–556, 2005.

[5] M. L. Puterman. *Markov Decision Processes—Discrete Stochastic Dynamic Programming*. John Wiley & Sons, Inc., 1994.

[6] N. Jong and P. Stone. Kernel-based models for reinforcement learning in continuous state spaces. In *Proceedings of the International Conference on Machine Learning—Workshop on Kernel Machines and Reinforcement Learning*, 2006.

[7] J. E. Cohen and U. G. Rothblum. Nonnegative ranks, decompositions and factorizations of nonnegative matrices. *Linear Algebra and its Applications*, 190:149–168, 1991.

[8] A. Cutler and L. Breiman. Archetypal analysis. *Technometrics*, 36(4):338–347, 1994.

[9] A. M. S. Barreto and M. D. Fragoso. Computing the stationary distribution of a finite Markov chain through stochastic factorization. *SIAM Journal on Matrix Analysis and Applications*. In press.

[10] J. Sorg and S. Singh. Transfer via soft homomorphisms. In *Autonomous Agents & Multiagent Systems / Agent Theories, Architectures, and Languages*, pages 741–748, 2009.

[11] S. A. Vavasis. On the complexity of nonnegative matrix factorization. *SIAM Journal on Optimization*, 20:1364–1377, 2009.

[12] W. Whitt. Approximations of dynamic programs, I. *Mathematics of Operations Research*, 3 (3):231–243, 1978.

[13] B. Ravindran. *An Algebraic Approach to Abstraction in Reinforcement Learning*. PhD thesis, University of Massachusetts, Amherst, MA, 2004.

[14] L. Kaufman and P. J. Rousseeuw. *Finding Groups in Data: an Introduction to Cluster Analysis*. John Wiley and Sons, 1990.

[15] R. S. Sutton. Generalization in reinforcement learning: Successful examples using sparse coarse coding. In *Advances in Neural Information Processing Systems*, volume 8, pages 1038–1044, 1996.

[16] F. J. Gomez. *Robust Non-linear Control Through Neuroevolution*. PhD thesis, The University of Texas at Austin, 2003.

[17] A. P. Wieland. Evolving neural network controllers for unstable systems. In *Proceedings of the International Joint Conference on Neural Networks*, volume 2, pages 667–673, 1991.

[18] F. Gomez, J. Schmidhuber, and R. Miikkulainen. Efficient non-linear control through neuroevolution. In *Proceedings of the 17th European Conference on Machine Learning*, pages 654–662, 2006.

[19] K. Bush, J. Pineau, and M. Avoli. Manifold embeddings for model-based reinforcement learning of neurostimulation policies. In *Proceedings of the ICML / UAI / COLT Workshop on Abstraction in Reinforcement Learning*, 2009.

[20] K. Bush and J. Pineau. Manifold embeddings for model-based reinforcement learning under partial observability. In *Advances in Neural Information Processing Systems*, volume 22, pages 189–197, 2009.

[21] K. Jerger and S. J. Schiff. Periodic pacing an in vitro epileptic focus. *Journal of Neurophysiology*, (2):876–879, 1995.

